# Learning the Dependency Structure of Latent Factors

**Yunlong He**[*]
Georgia Institute of Technology
heyunlong@gatech.edu

**Yanjun Qi**
NEC Labs America
yanjun@nec-labs.com

**Koray Kavukcuoglu**
NEC Labs America
koray@nec-labs.com

**Haesun Park**[*]
Georgia Institute of Technology
hpark@cc.gatech.edu

## Abstract

In this paper, we study latent factor models with dependency structure in the latent space. We propose a general learning framework which induces sparsity on the undirected graphical model imposed on the vector of latent factors. A novel latent factor model SLFA is then proposed as a matrix factorization problem with a special regularization term that encourages collaborative reconstruction. The main benefit (novelty) of the model is that we can simultaneously learn the lower-dimensional representation for data and model the pairwise relationships between latent factors explicitly. An on-line learning algorithm is devised to make the model feasible for large-scale learning problems. Experimental results on two synthetic data and two real-world data sets demonstrate that pairwise relationships and latent factors learned by our model provide a more structured way of exploring high-dimensional data, and the learned representations achieve the state-of-the-art classification performance.

## 1 Introduction

Data samples described in high-dimensional feature spaces are encountered in many important areas. To enable the efficient processing of large data collections, latent factor models (LFMs) have been proposed to find concise descriptions of the members of a data collection. A random vector $\mathbf{x} \in \mathbb{R}^M$ is assumed to be generated by a linear combination of a set of basis vectors, i.e.,

$$\mathbf{x} = \mathbf{B}s + \epsilon = \mathbf{B}_1 s_1 + \mathbf{B}_2 s_2 + \cdots + \mathbf{B}_K s_K + \epsilon \tag{1}$$

where $\mathbf{B} = [\mathbf{B}_1, \ldots, \mathbf{B}_K]$ stores the set of unknown basis vectors and $\epsilon$ describes noise. The $i$-th "factor" $s_i$ ($i \in \{1, ..., K\}$) denotes the $i$-th variable in the vector $\mathbf{s}$.

In this paper, we consider the problem of learning hidden dependency structure of latent factors in complex data sets. Our goal includes two main aspects: (1) to learn the interpretable lower-dimensional representations hidden in a set of data samples, and (2) to simultaneously model the pairwise interaction of latent factors. It is difficult to achieve both aspects at the same time using existing models. The statistical structure captured by LFM methods, such as Principal Component Analysis (PCA) are limited in interpretability, due to their anti-correlation assumption on the latent factors. For example, when a face image is represented as a linear super-position of PCA bases with uncorrelated coefficients learned by PCA, there exist complex cancellations between the basis images [14]. Methods that theoretically assume independence of components like ICA [10] or sparse coding [15] fail to generate independent representations in practice. Notable results in [13, 17] have shown that the coefficients of linear features for natural images are never independent.

---

[*]The work of these authors was supported in part by the National Science Foundation grant CCF-0808863. Any opinions, findings and conclusions or recommendations expressed in this material are those of the authors and do not necessarily reect the views of the National Science Foundation.

Instead of imposing this unrealistic assumption, more recent works [18, 25, 27] propose to allow correlated latent factors, which shows to be helpful in obtaining better performance on various tasks. However, the graphical structure of latent factors (i.e., conditional dependence/independence) is not considered in these works. Particularly, the sparse structure of the latent factor network is often preferred but has been never been explicitly explored in the learning process [2, 8, 23]. For example, when mining the enormous on-line news-text documents, a method discovering semantically meaningful latent topics and a concise graph connecting the topics will greatly assist intelligent browsing, organizing and accessing of these documents.

The main contribution in this paper is a general LFM method that models the pairwise relationships between latent factors by sparse graphical models. By introducing a generalized Tikhonov regularization, we enforce the interaction of latent factors to have an influence on learning latent factors and basis vectors. As a result, we learn meaningful latent factors and simultaneously obtain a graph where the nodes represent hidden groups and the edges represent their pairwise relationships. This graphical representation helps us analyze collections of complex data samples in a much more structured and organized way. The latent representations of data samples obtained from our model capture deeper signals hidden in the data which produce the useful features for discriminative task and in-depth analysis, e.g. our model achieves a state-of-the-art performance on classifying cancer samples in our experiment.

## 2  Methods

### 2.1  Sparse Undirected Graphical Model of Latent Factors: A General Formulation

Following [4, 16], our framework considers data samples drawn from the exponential family of distributions, i.e.,

$$p(\mathbf{x}|\eta) = h(\mathbf{x})exp(\eta^\intercal T(\mathbf{x}) - A(\eta)), \tag{2}$$

where sufficient statitic $T(\mathbf{x}) \in \mathbb{R}^M$, $\eta \in \mathbb{R}^M$ represents the natural parameter for the model, $T(\mathbf{x})$, $h(\mathbf{x})$ and $A(\eta)$ are known functions defining a particular member of the exponential family. This family includes most of the common distributions, like normal, Dirichlet, multinomial, Poisson, and many others.

To learn the hidden factors for generating $\mathbf{x}$, the natural parameter $\eta$ is assumed to be represented by a linear combination of basis vectors, i.e.,

$$\eta = \mathbf{B}\mathbf{s}, \tag{3}$$

where $\mathbf{B} = [\mathbf{B}_1, \dots, \mathbf{B}_K]$ is the basis matrix. To model the pairwise interaction between latent factors, we introduce a pairwise Markov Random Field (MRF) prior on the vector of factors $\mathbf{s} \in \mathbb{R}^K$:

$$p(\mathbf{s}|\mu, \mathbf{\Theta}) = \frac{1}{Z(\mu, \mathbf{\Theta})} exp(-\sum_{i=1}^{K} \mu_i s_i - \frac{1}{2} \sum_{i=1}^{K} \sum_{j=1}^{K} \theta_{ij} s_i s_j) \tag{4}$$

with parameter $\mu = [\mu_i]$, symmetric $\mathbf{\Theta} = [\theta_{ij}]$, and partition function $Z(\mu, \mathbf{\Theta})$ which normalizes the distribution. The classic Ising model and Gaussian graphical model are two special cases of the above MRF. Let $G = (V, E)$ denote a graph with $K$ nodes, corresponding to the $K$ latent factors $\{s_1, \dots, s_K\}$, and with edge set

$$E = \{(i, j) \in V \times V : \theta_{ij} \neq 0\}. \tag{5}$$

Since $\theta_{ij} = 0$ indicates that latent factor $s_i$ and latent factor $s_j$ are conditionally independent given other latent factors, the graph $G$ presents an illustrative view of the statistical dependencies between latent factors.

With such a hierarchical and flexible model, there would be significant risk of over-fitting, especially when we consider all possible interactions between $K$ latent factors. Therefore, regularization has to be introduced for better generalization property of the model. As we will see in subsection 3, regularization is also necessary from the perspective of avoiding ill-posed optimization problem. The regularization technique we use is to introduce a sparsity-inducing prior for $\mathbf{\Theta}$:

$$p(\mathbf{\Theta}) \propto \exp(-\frac{1}{2}\rho\|\mathbf{\Theta}\|_1), \tag{6}$$

where $\rho$ is a positive hyper-parameter and $\|\mathbf{\Theta}\|_1 := \sum_i \sum_j |\theta_{ij}|$. We aim to achieve two goals when designing such a prior distribution: (1) in practice irrelevant latent factors are not supposed to be conditionally dependent and hence a concise graphical structure between latent factors is preferred in many applications such as topic mining and image feature learning, and (2) in contrast to $L_0$ regularization which is the number of non-zero components, we obtain a convex subproblem of $\mathbf{\Theta}$, that can be efficiently solved by utilizing the recently developed convex optimization techniques.

## 2.2 Learning Algorithm

We consider the posterior distribution of parameters, which is proportional to the product of data likelihood and the prior distributions:

$$h(\mathbf{x})exp\{\mathbf{s}^\mathsf{T}\mathbf{B}^\mathsf{T}T(\mathbf{x}) - A(\mathbf{Bs})\} \times \frac{1}{Z(\mu, \mathbf{\Theta})}exp(-\mu^\mathsf{T}\mathbf{s} - \frac{1}{2}\mathbf{s}^\mathsf{T}\mathbf{\Theta s}) \times exp(-\frac{1}{2}\rho\|\mathbf{\Theta}\|_1). \quad (7)$$

Given a set of data observations $\{\mathbf{x}^{(1)}, \ldots, \mathbf{x}^{(N)}\}$, the Maximum a Posteriori (MAP) estimates of the basis matrix $\mathbf{B}$, the latent factors in $\mathbf{S} = [\mathbf{s}^{(1)}, \ldots, \mathbf{s}^{(N)}]$ and the parameters $\{\mu, \mathbf{\Theta}\}$ of the latent factor network are therefore the solution of the following problem:

$$\min_{\mathbf{B}, \mathbf{S}, \mathbf{\Theta}} \frac{1}{N}\sum_i \{-\log h(\mathbf{x}^{(i)}) + A(\mathbf{Bs}^{(i)}) - \mathbf{s}^{(i)\mathsf{T}}\mathbf{B}^\mathsf{T}T(\mathbf{x}^{(i)})\}$$

$$+ \log Z(\mu, \mathbf{\Theta}) + \frac{1}{N}\mu^\mathsf{T}\mathbf{S}\mathbf{1}_N + \frac{1}{2N}tr(\mathbf{S}^\mathsf{T}\mathbf{\Theta S}) + \frac{1}{2}\rho\|\mathbf{\Theta}\|_1$$

$$\text{s.t. } \mathbf{B} \geq \mathbf{0}, \|\mathbf{B}_k\|_2 \leq 1, k = 1, \ldots, K, \quad (8)$$

where additional constrains $\mathbf{B} \geq \mathbf{0}$ and $\|\mathbf{B}_k\|_2 \leq 1$ are introduced for the identifiability of the model.

The objective function in Eq. (8) is not convex with respect to all three unknowns ($\mathbf{B}$, $\mathbf{S}$ and $\mathbf{\Theta}$) together. Therefore, a good algorithm in general exhibits convergence behavior to a stationary point and we can use Block Coordinate Descent algorithm [1] to iteratively update $\mathbf{B}$, $\mathbf{S}$ and $\mathbf{\Theta}$ as follows:

**while** not convergent **do**

For $i = 1, \ldots, N$, solve

$$\min_{s^{(i)}} -\log h(\mathbf{x}^{(i)}) + A(\mathbf{Bs}^{(i)}) - \mathbf{s}^{(i)T}\mathbf{B}^\mathsf{T}T(\mathbf{x}^{(i)}) + \mu^\mathsf{T}\mathbf{s}^{(i)} + \frac{1}{2}\mathbf{s}^{(i)T}\mathbf{\Theta s}^{(i)} \quad (9)$$

Solve

$$\min_{\mathbf{B} \geq \mathbf{0}, \|\mathbf{B}_k\|_2 \leq 1} \sum_i \{-\log h(\mathbf{x}^{(i)}) + A(\mathbf{Bs}^{(i)}) - \mathbf{s}^{(i)T}\mathbf{B}^\mathsf{T}T(\mathbf{x}^{(i)})\} \quad (10)$$

Solve

$$\min_{\mu, \mathbf{\Theta}} \log Z(\mu, \mathbf{\Theta}) + \frac{1}{N}\mu^\mathsf{T}\mathbf{S}\mathbf{1}_N + \frac{1}{2N}tr(\mathbf{S}^\mathsf{T}\mathbf{\Theta S}) + \frac{1}{2}\rho\|\mathbf{\Theta}\|_1 \quad (11)$$

**end do**

Since $p(\mathbf{x}|\eta)$ is in the exponential family, the subproblem (10) with respect to $\mathbf{B}$ is convex and smooth with simple constraints, for which quasi-Newton methods such as projected L-BFGS [22] are among the most efficient methods. Subproblem (9) is easy to solve for real-valued $s^{(i)}$ but generally hard when the latent factors only admit discrete values. For example for $\mathbf{s} \in \{0, 1\}^K$ and Gaussian $p(\mathbf{x}|\eta)$, subproblem (9) is a 0-1 quadratic programming problem and we can resort to SDP based Branch and Bound algorithms [20] to solve it in a reasonable time. The subproblem (11) is minimizing the sum of a differentiable convex function and an $L_1$ regularization term, for which a few recently developed methods can be very efficient, such as variants of ADMM [6]. For the cases of discrete $\mathbf{s}$ with large $K$ (usually $K << M$), evaluation of the partition function $Z(\mu, \mathbf{\Theta})$ during the iterations is $\sharp$P-hard and Schmidt [21] discusses methods to solve the pseudo-likelihood approximation of (11).

## 3 A Special Case: Structured Latent Factor Analysis

From this section on, we consider a special case of the learning problem in Eq. (8) when $\mathbf{x}$ follows a multivariate normal distribution and $\mathbf{s}$ follows a sparse Gaussian graphical model (SGGM). We name our model under this default setting as "structured latent factor analysis" (SLFA) and compare it to related works. Assume $p(\mathbf{x}|\eta) = (2\pi)^{-M/2}exp(-\frac{1}{2\sigma^2}\|\mathbf{x} - \eta\|^2)$ and $\mathbf{s} \sim N(\mu, \mathbf{\Phi}^{-1})$, with

sparse precision matrix $\mathbf{\Phi}$ (inverse covariance). For simplicity we assume the given data matrix $\mathbf{X} = [\mathbf{x}^{(1)}, \ldots, \mathbf{x}^{(N)}]$ is centered and set $\mu = 0$. Then the objective function in Eq. (8) becomes

$$\min_{\mathbf{B},\mathbf{S},\mathbf{\Phi}} \frac{1}{N}\|\mathbf{X} - \mathbf{BS}\|_F^2 + \sigma^2(\frac{1}{N}\text{tr}(\mathbf{S}^\mathsf{T}\mathbf{\Phi}\mathbf{S}) - \log\det(\mathbf{\Phi}) + \rho\|\mathbf{\Phi}\|_1)$$
$$\text{s.t. } \mathbf{B} \geq \mathbf{0}, \|\mathbf{B}_k\|_2 \leq 1, k = 1, \ldots, K, \mathbf{\Phi} \succcurlyeq 0. \tag{12}$$

If $\mathbf{\Phi}$ is fixed, the problem in Eq. (12) is a matrix factorization method with generalized Tikhonov regularization: $trace(\mathbf{S}^\mathsf{T}\mathbf{\Phi}\mathbf{S})$. If $\Phi_{i,j} > 0$, minimizing the objective function will avoid $s_i$ and $s_j$ to be simultaneously large, and we say the $i$-th factor and the $j$-th factor are negatively related. If $\Phi_{i,j} < 0$, the solution is likely to have $s_i$ and $s_j$ of the same sign, and we say the $i$-th factor and the $j$-th factor are positively related. If $\Phi_{i,j} = 0$, the regularization doesn't induce interaction between $s_i$ and $s_j$ in the objective function. Therefore, this regularization term makes SLFA produce a collaborative reconstruction based on the conditional dependencies between latent factors. On one hand, the collaborative nature makes SLFA capture deeper statistical structure hidden in the data set, compared to the matrix factorization problem with the Tikhonov regularization $\|\mathbf{S}\|_F^2$ or sparse coding with the sparsity-inducing regularization such as $\|\mathbf{S}\|_1$. On the other hand, SLFA encourages sparse interactions which is very different from previous works such as correlated topic Model [2] and latent Gaussian model [18], where the latent factors are densely related.

**An On-line Algorithm For Learning SLFA:** The convex subproblem

$$\min_{\mathbf{\Phi}\succcurlyeq 0} \frac{1}{N}\text{tr}(\mathbf{S}^\mathsf{T}\mathbf{\Phi}\mathbf{S}) - \log\det(\mathbf{\Phi}) + \rho\|\mathbf{\Phi}\|_1 \tag{13}$$

can be efficiently solved by a recent quadratic approximation method in [9]. For subproblem of $S$ we have closed-form solution
$$\mathbf{S} = (\mathbf{B}^\mathsf{T}\mathbf{B} + \sigma^2\mathbf{\Phi})^{-1}\mathbf{X}.$$

Moreover, considering that many modern high-dimensional data sets include a large number of data observations (e.g. text articles from web-news), we propose an online algorithm for learning SLFA on larger data sets. As summarized in Algorithm 1, at each iteration, we randomly fetch a mini-batch of observations simultaneously, compute their latent factor vector $\mathbf{s}$. Then the latent factor vectors are used to update the basis matrix $\mathbf{B}$ in stochastic gradient descent fashion with projections on the constraint set. Lastly we update the precision matrix $\mathbf{\Phi}$.

---

**Algorithm 1** An on-line algorithm for learning SLFA.

---

**Input:** $\mathbf{X} = [\mathbf{x}^{(1)}, \ldots, \mathbf{x}^{(N)}]$, initial guess of basis matrix $\mathbf{B}$, initial precision matrix $\mathbf{\Phi} = I$, number of iterations $T$, parameters $\sigma^2$ and $\rho$, step-size $\gamma$, mini-batch size $N'$.

- **for** $t = 1$ **to** $T$
  - Draw $N'$ observations randomly from $\mathbf{X} = [\mathbf{x}^{(1)}, \ldots, \mathbf{x}^{(N)}]$ to form the matrix $\mathbf{X}_{batch}$.
  - Compute the latent factor vectors $\mathbf{S}_{batch} = (\mathbf{B}^\mathsf{T}\mathbf{B} + \sigma^2\mathbf{\Phi})^{-1}\mathbf{X}_{batch}$.
  - Update the basis matrix $\mathbf{B}$ using a gradient descent step:
    $\mathbf{B} \leftarrow \mathbf{B} - \frac{\gamma}{N'}[\mathbf{BS}_{batch} - \mathbf{X}_{batch}]\mathbf{S}_{batch}^\mathsf{T}$.
  - Project columns of $\mathbf{B}$ to the first orthant and the unit ball, i.e., $\mathbf{B} \geq \mathbf{0}$ and $\|\mathbf{B}_i\| \leq 1$.
  - Solve the subproblem (13) to update the sparse inverse covariance matrix $\mathbf{\Phi}$ using all available latent factor vectors in $\mathbf{S}$.
- **end for**

---

**Parameter Selection:** The hyper-parameter $\rho$ controls the sparsity of $\mathbf{\Phi}$. A large $\rho$ will result in a diagonal precision matrix $\mathbf{\Phi}$, indicating that the latent factors are conditionally independent. As $\rho \to 0$, $\mathbf{\Phi}$ becomes denser. However, if we set $\rho = 0$, the subproblem with respect to $\mathbf{\Phi}$ has a closed form solution $\mathbf{\Phi} = (\frac{1}{N}\mathbf{SS}^\mathsf{T})^{-1}$, i.e., inverse sample covariance matrix. Plugging it back to the Eq. (12), we have
$$\min_{\mathbf{B},\mathbf{S}} \frac{1}{N}\|\mathbf{X} - \mathbf{BS}\|_F^2 + \sigma^2\log det(\frac{1}{N}\mathbf{SS}^\mathsf{T}),$$

which doesn't have a lower bound. Therefore the regularization is necessary and we choose positive values for $\rho$ in the experiments. For supervised tasks, we use cross-validation to choose the proper

value of $\rho$ that optimizes the evaluation rule on validation set. For unsupervised applications, we combine the BIC criterion in [28], with our model to obtain the following criterion:

$$\rho^* = \min_\rho \frac{1}{N}\|\mathbf{X} - \mathbf{B}(\rho)\mathbf{S}(\rho)\|_F^2 + \sigma^2 \left( \frac{1}{N}\mathrm{tr}(\mathbf{S}(\rho)^\mathsf{T}\boldsymbol{\Phi}(\rho)\mathbf{S}(\rho)) - \log\det(\boldsymbol{\Phi}(\rho)) + \frac{\log N}{N}\|\boldsymbol{\Phi}(\rho)\|_0 \right),$$

where $\mathbf{B}(\rho)$, $\mathbf{S}(\rho)$ and $\boldsymbol{\Phi}(\rho)$ and learned from (12) with parameter $\rho$. Alternatively, for visual analysis of latent factors, we can select multiple values of $\rho$ to obtain $\boldsymbol{\Phi}$ with desired sparsity.

**Relationship to Sparse Gaussian Graphical Model:** We can also see SLFA as a generalization of sparse Gaussian graphical model. In fact, if the reduced dimension $K = M$, the problem (12) has trivial solution $\mathbf{B} = \mathbf{I}$ and $\mathbf{S} = \mathbf{X}$, and the problem becomes the same as (13). When $K < M$, the subproblem with respect to $\mathbf{s}$ has solution $\mathbf{s} = (\mathbf{B}^\mathsf{T}\mathbf{B} + \sigma^2\boldsymbol{\Phi})^{-1}\mathbf{x}$. Therefore, lower dimensional random vector $\mathbf{s}$ has less variables among which each variable is a linear combination of the original variables of $\mathbf{x}$ with the combination weights stored in $\mathbf{W} = (\mathbf{B}^\mathsf{T}\mathbf{B} + \sigma^2\boldsymbol{\Phi})^{-1}$. In this sense, SLFA could be seen as the sparse Gaussian graphical model of $\mathbf{s} = \mathbf{W}\mathbf{x}$, i.e. it generalizes the concept from the original (totally $N$) variables to the merged (totally $K$) group variables.

A few recent efforts [3, 24] also combined the model of SGGM and with latent factor models. For example, "Kronecker GLasso" in [24] performs a joint learning of row and column covariances for matrix-variate Gaussian models. Different from our SLFA, these methods still aim at modeling the interaction between the original features and doesn't consider interaction in the latent factor space. Instead, SLFA is a hierarchical model and the learned pairwise relationships are on the latent factor level. If we apply both SLFA and Kronecker GLasso on a text corpus where each document is represented by a $50,000$ sparse vector and number of latent factors (topics) are fixed as $50$, then Kronecker GLasso will produce a precision matrix of dimension $50,000 \times 50,000$ and a corresponding sparse graph of $50,000$ nodes. SLFA, however, can dramatically reduce the problem to learning a $50 \times 50$ sparse precision matrix and the corresponding graph of $50$ nodes.

**Relationship to other works:** Sparse coding [19] can be modeled as:

$$\min_{\mathbf{B},\mathbf{S}} \frac{1}{2}\|\mathbf{X} - \mathbf{B}\mathbf{S}\|_F^2 + \lambda\|\mathbf{S}\|_1. \tag{14}$$

For many high-dimensional data sets such as text in natural languages, the input data is already very sparse or high dimensional. Thus, sparse coding is not easily applicable. Intuitively, sparse coding based works (such as [7]) try to remove the redundancy in the representation of data while SLFA encourages a (sparse) collaborative reconstruction of the data from the latent bases.

Recently, Jenatton et al. [12] proposed a method that can learn latent factors with given tree structure. The optimization problem in Jenatton et al., 2010 is a penalized matrix factorization problem similar to our Eq. (12) and Eq. (14), but uses a different regularization term which imposes the overlapped group sparsity of factors. Differently, SLFA can learn a more general graphical structure among latent factors and doesn't assume that data sample maps to a sparse combination of basis vectors.

The model of SLFA has similar hierarchy with correlated topic model [2] and latent Gaussian model [18]. Besides the key difference of sparsity, SLFA directly use precision matrix to learn latent factor networks while the other two works learn the covariance matrix by Bayesian methods.

## 4 Experiments

In this section, we conduct experiments on both synthetic and real world data sets to show that: (1) SLFA recovers latent basis vectors and finds the pairwise relationships of latent factors, (2) SLFA generates useful features for various tasks such as images analysis, topic visualization and microarray analysis.

### 4.1 Synthetic Data I: Four Different Graphical Relationships

The first experiment uses randomly generated synthetic data with different graphical structures of latent factors. It aims to test if SLFA can find true latent factors and the true relationships among latent factors and to study the effect of the parameter $\rho$ on the results. We use four special cases of Sparse Gaussian Graphical Model to generate the latent factors. The underlying graph is either a ring, a grid, a tree or a random sparse graph, which are shown in Figure 1. A sparse positive

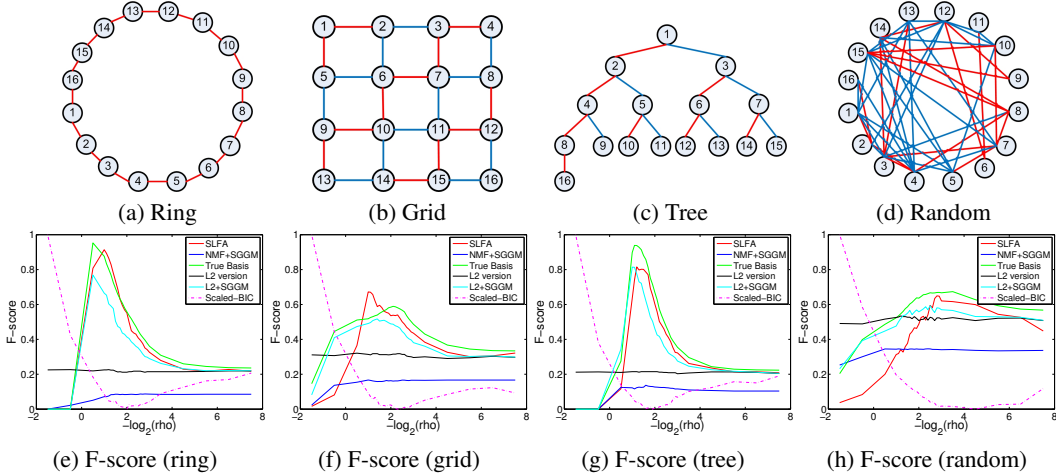

(a) Ring     (b) Grid     (c) Tree     (d) Random

(e) F-score (ring)    (f) F-score (grid)    (g) F-score (tree)    (h) F-score (random)

Figure 1: Recovering structured latent factors from data. On the upper row are four different underlying graphical model of latent factors. Red edge means the two latent factors are positively related ($\Phi_{ij}^* < 0$), blue edge implies the two latent factors are negatively related ($\Phi_{ij}^* > 0$). On the lower row are the plots of F-score vs. $\rho$ for four settings. We can observe that SLFA (red lines) is as good as an oracle method (True Basis, green lines). The pink dash lines of BIC score (scaled to $[0,1]$) demonstrate that the parameter selection method works well.

definite matrix $\boldsymbol{\Phi}^* \in \mathbb{R}^{10\times 10}$ is constructed based on the graph of SGGM. Then we sample 200 Gaussian random vectors, $\mathbf{s}^{(1)}, \ldots, \mathbf{s}^{(200)} \in \mathbb{R}^{10}$, with precision matrix $\boldsymbol{\Phi}^*$. A set of vectors $\mathbf{B}^* \in \mathbb{R}^{500\times 10}$ is randomly generated with normal distribution and then filtered by a sigmoid function $f(b) = \frac{1}{1+e^{-100b}}$ such that most components of $\mathbf{B}^*$ are close to either 0 or 1. $\mathbf{B}_1, \mathbf{B}_2, \ldots, \mathbf{B}_{10}$ are then normalized as basis vectors. Finally, the synthetic data points are generated by $\mathbf{x}^{(i)} = \mathbf{B}\mathbf{s}^{(i)} + 0.1\epsilon_i, i = 1, \ldots, 200$, where $\epsilon_i \sim N(\mathbf{0}, \mathbf{I})$.

We compare SLFA to other four methods for learning the basis matrix $\mathbf{B}$ and the precision matrix $\boldsymbol{\Phi}$ from the data. The first one is NMF, where we learn nonnegative basis $\mathbf{B}$ from the data and then learn the sparse precision matrix $\boldsymbol{\Phi}$ for the corresponding factor vectors (non nonnegative constraint on factors) by SGGM. The second one is an ideal case where we have the "oracle" of the true basis $\mathbf{B}^*$, then after fit the data to be true basis we learn the sparse precision matrix $\boldsymbol{\Phi}$ by SGGM. The third one is named $L_2$ version of SLFA as we replace the $L_1$ regularization of $\boldsymbol{\Phi}$ by a Frobenius norm regularization. The fourth method first applies L2 version of SLFA and then learns $\boldsymbol{\Phi}$ by SGGM. In all cases except the oracle method, we have a non-convex problem so that after we obtain the learned basis vectors we use Hungarian algorithm to align them to with the true basis vectors based on the cosine similarity. We compute the precision and recall rates for recovering the relationship between latent factors by comparing the learned $\boldsymbol{\Phi}$ with the true precision matrix $\boldsymbol{\Phi}^*$.

We plot F-score based on the precision and recall rates averaged over 10 experiments. According to Figure 1, when $\rho$ is large, the estimated $\boldsymbol{\Phi}$ is diagonal so that recall rate is 0. As $\rho$ becomes smaller, more nonzero elements appear in the estimated $\boldsymbol{\Phi}$ and both the recall and precision rate of "positive/negative relationship" get increased. When $\rho$ is small enough, the recovered $\boldsymbol{\Phi}$ becomes denser and may not even recover the "positive/negative relationship" correctly. We can see that for all four cases, our proposed method SLFA is as good as the "oracle" method at recovering the pairwise relationship between latent factors. NMF most probably fails to find the right basis since it does consider any higher level information about the interactions between basis elements, hence SGGM can't find meaningful relationship between the factors obtained from NMF. $L_2$ version of SLFA also has poor F-score since it can't recover the sparse structure. Since latent factors have dense interactions in $L_2$ version of SLFA, combining it with a postprocessing by SGGM improves the performance significantly, however it still performs worse compared to SLFA. This experiment also confirms that the idea of performing an integrated learning of the bases together with a regularized precision matrix is essential for recovering the true structure in the data.

## 4.2 Synthetic Data II: Parts-based Images

The second experiment also utilizes a simulated data set based on images to compare SLFA with popular latent factor models. We set up an experiment by generating 15000 images of *"bugs"*, each

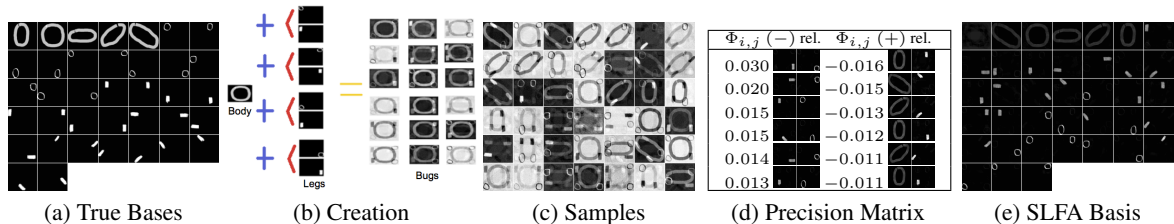

| | (a) True Bases | (b) Creation | (c) Samples | (d) Precision Matrix | (e) SLFA Basis |

Figure 2: Table (e) shows the $\Phi(i,j)$ values and corresponding $B_i$ and $B_j$ elements learned by SLFA for the six highest and and six lowest entries in $\Phi$. For $\Phi(i,j) > 0$, $B_i$ and $B_j$ are negatively related (exclusive), for $\Phi(i,j) < 0$, $B_i$ and $B_j$ are positively related (supportive).

of which is essentially a linear combination of five latent parts shown in Figure 2a. Given 37 basis images, we first randomly select one of the five big circles as the body of the "bugs". Each shape of body is associated with four positions where the legs of the bug is located. We then randomly pick 4 legs from its associated set of 4 small circles and 4 small squares. However, for each leg, circle and square are exclusive of each other. We combine the selected five latent parts with random coefficients that are sampled from the uniform distribution and multiplied by $-1$ with probability $0.5$. Finally, we add a randomly selected basis with small random coefficients plus Gaussian random noise to the image to introduce the noise and confusion in the data set. A few examples of the bug image samples created by the above strategy are shown in Figure 2c. The generating process (Figure 2b) indicates positive relationship between one type of body and its associates legs, as well as negative relationship between the pair of circle and square that is located at the same position.

Using SLFA and other two baseline algorithms, PCA and NMF, we learn a set of latent bases and compare the result of three methods in Figures 2e. We can see that the basis images generated by SLFA is almost exactly same as the true latent bases. This is due to the fact that SLFA accounts for the sparse interaction between factors in the joint optimization problem and encourages collaborative reconstruction. NMF basis (shown in supplementary material due to space considerations) in this case also turns out to be similar to true basis, however, one can still observe that many components contain mixed structures since it can not capture the true data generation process. The bases learned by PCA (also shown in supp. material) is not interpretable as expected.

More importantly, SLFA provides the convenience of analyzing the relationship between the bases using the precision matrix $\Phi$. In Figure 2d, we analyze the relational structure learned in the precision matrix $\Phi$. The most negatively related (exclusive) pairs (the $i$ and $j$ entries with highest positive entries in $\Phi$) are circular and square legs which conforms fully to the generation process, since only one of them is chosen for any given location. Accordingly, the most positively related pairs are a body shape and one of its associated legs since every bug has a body and four legs with fixed positions.

### 4.3 Real Data I: NIPS Documents

In this section, we apply SLFA to the NIPS corpus[1] which contains 1740 abstracts from the NIPS Conferences $1-12$ for the purpose of topic/content modeling. SLFA is used to organize and visualize the relationship between the structured topics. SLFA is applied on the 13649 dimensional tf-idf feature vector which is normalized to have the unit norm. We fix the number of topics to be 40 and tune the parameters $\sigma$ and $\rho$ to obtain $\Phi$ with a proper sparsity for the visualization task. In figure 3, we plot a graph of topics (standing-alone topics removed) with positive interaction between each other and present the top 5 keywords for each topic. For example, the topic at the top is about general notions in many learning algorithms and acts as the hub point of the graph. more specific words that are relevant to a particular learning algorithm or a more specialized topic of interest. It is obvious that SLFA not only extracts the underlying topics, but is also able to capture the (de)correlations between topics. For example, on the far left, the topic related to cells is connected to *"motion, velocity, ..."*, *"objects, image,..."* and *"spike, neurons, ..."* nodes. This subgraph clearly represents a few topics in computer vision and neuroscience. The node on the far right containing *"robot, planning, ..."* is connected to the node with *"controller, control, ..."* which represents a robotics related topic-cluster. It is also interesting to note that SLFA can obtain a graph of negatively related topics(shown in supplementary material). One can see that closely related topics tend to exclude each other.

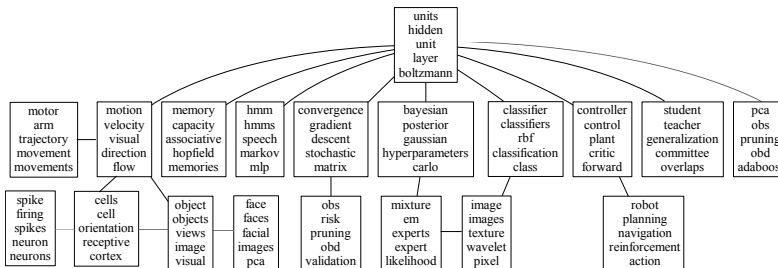

Figure 3: Positively related topics (learned by SLFA) discovered from NIPS text corpus. Each edge corresponds to a negative element in the sparse precision matrix $\mathbf{\Phi}$.

| SLFA | Lasso-overlapped-group | Lasso | SVM | PCA |
|---|---|---|---|---|
| $\mathbf{34.22 \pm 2.58}$ | $35.31 \pm 2.05$ | $36.42 \pm 2.50$ | $36.93 \pm 2.54$ | $36.85 \pm 3.02$ |

Table 1: Cross-validation error rate (average and standard deviation) by different methods on Gene Micro-array data. SLFA performs best and even better than Lasso-overlapped-group (t-test at significance level 0.02), which takes advantage of external information ($42,594$ known edges between gene variables from another biological resource).

### 4.4   Real Data II: Gene Microarray Data for Cancer Classification

Next, we test our model on a classification task which uses breast cancer microarray data set obtained from [11]. This data set contains the gene expression values of $8,141$ genes for $295$ breast cancer tumor samples. The task is to classify the tumor samples into two classes (with $78$ metastatic and $217$ non-metastatic).

Using the classification error rates as the metric, we compare totally five methods, including Lasso [26], Lasso-overlapped-group [11], linear SVM classifier [5], PCA with linear SVM classifier and SLFA with linear SVM classifier. Lasso-overlapped-group, which is a logistic regression approach with the graph-guided sparsity enforced, uses a known biological network as the graphical (overlapped group) regularization on the lasso regression. The other methods, including SLFA, do not use this extra supervised information. We run $10$-fold cross validation and use the averaged error rate to indicate the predictive performance of different methods. The test is repeated $50$ times and each time all methods use the same split of training and validation sets.

The averaged cross-validation error rate is shown in Table 1. We can observe that SLFA ($K = 100$) has lower error rates than other methods, including Lasso, SVM and PCA. Compared to the method of Lasso-overlapped-group [11] which constructs the regularization from external information ($42,594$ known edges as prior knowledge), our method based on SLFA performs better, even though it does not utilize any extra evidence. This is a strong evidence which indicates that SLFA can extract deeper structural information hidden in the data. Indeed, genes naturally act in the form of functional modules (gene groups) to carry out specific functions. Gene groups that usually correspond to biological processes or pathways, exhibit diverse pairwise dependency relationships among each other. SLFA discovers these relationships while learning the latent representation of each data sample at the same time. That is why its learned lower-dimensional representation captures more fundamental and strong signals, and achieves the state-of-art classification performance. The learned structural information and latent gene groups also get confirmed by the biological function analysis in supplementary document.

## 5   Conclusion

In this paper we have introduced a novel structured latent factor model that simultaneously learns latent factors and their pairwise relationships. The model is formulated to represent data drawn from the general exponential family of distributions. The learned sparse interaction between latent factors is crucial for understanding complex data sets and to visually analyze them. SLFA model is also a hierarchical extension of Sparse Gaussian Graphical Model by generalizing the application of precision matrix from the original variable space to the latent factor space and optimizing the bases together with the precision matrix simultaneously. We have also provided an efficient online learning algorithm that can scale SLFA training to large-scale datasets and showed that SLFA not only can predict the true basis and structured relationshop between bases, but also it can achieve state-of-the-art results in challenging biological classification task.

## Footnotes

[1]http://cs.nyu.edu/ roweis/data.html

# References

[1] Bertsekas, D.: Nonlinear programming. Athena Scientific Belmont, MA (1999)

[2] Blei, D., Lafferty, J.: Correlated topic models. Advances in Neural Information Processing Systems (2006)

[3] Chandrasekaran, V., Parrilo, P., Willsky, A.: Latent variable graphical model selection via convex optimization. Arxiv preprint arXiv:1008.1290 (2010)

[4] Collins, M., Dasgupta, S., Schapire, R.: A generalization of principal component analysis to the exponential family. Advances in neural information processing systems (2002)

[5] Fan, R., Chang, K., Hsieh, C., Wang, X., Lin, C.: Liblinear: A library for large linear classification. JMLR (2008)

[6] Goldfarb, D., Ma, S., Scheinberg, K.: Fast alternating linearization methods for minimizing the sum of two convex functions. Arxiv preprint arXiv:0912.4571 (2009)

[7] Gregor, K., Szlam, A., LeCun, Y.: Structured sparse coding via lateral inhibition. Advances in Neural Information Processing Systems **24** (2011)

[8] Hinton, G., Osindero, S., Bao, K.: Learning causally linked markov random fields. In: AI & Statistics (2005)

[9] Hsieh, C., Sustik, M., Ravikumar, P., Dhillon, I.: Sparse inverse covariance matrix estimation using quadratic approximation. Advances in Neural Information Processing Systems (NIPS) **24** (2011)

[10] Hyvärinen, A., Hurri, J., Hoyer, P.: Independent component analysis. Natural Image Statistics (2009)

[11] Jacob, L., Obozinski, G., Vert, J.: Group lasso with overlap and graph lasso. Proceedings of the 26th Annual International Conference on Machine Learning (2009)

[12] Jenatton, R., Mairal, J., Obozinski, G., Bach, F.: Proximal methods for sparse hierarchical dictionary learning. Proceedings of the International Conference on Machine Learning (2010)

[13] Karklin, Y., Lewicki, M.S.: Emergence of complex cell properties by learning to generalize in natural scenes. Nature (2009)

[14] Lee, D., Seung, H.: Learning the parts of objects by non-negative matrix factorization. Nature (1999)

[15] Lee, H., Battle, A., Raina, R., Ng, A.: Efficient sparse coding algorithms. Advances in neural information processing systems (2007)

[16] Lee, H., Raina, R., Teichman, A., Ng, A.: Exponential family sparse coding with applications to self-taught learning. Proceedings of the 21st international jont conference on Artifical intelligence (2009)

[17] Lyu, S., Simoncelli, E.: Nonlinear extraction of independent components of natural images using radial gaussianization. Neural computation (2009)

[18] Murray, I., Adams, R.: Slice sampling covariance hyperparameters of latent gaussian models. Arxiv preprint arXiv:1006.0868 (2010)

[19] Olshausen, B., et al.: Emergence of simple-cell receptive field properties by learning a sparse code for natural images. Nature (1996)

[20] Rendl, F., Rinaldi, G., Wiegele, A.: Solving Max-Cut to optimality by intersecting semidefinite and polyhedral relaxations. Math. Programming **121**(2), 307 (2010)

[21] Schmidt, M.: Graphical model structure learning with l1-regularization. Ph.D. thesis, UNIVERSITY OF BRITISH COLUMBIA (2010)

[22] Schmidt, M., Van Den Berg, E., Friedlander, M., Murphy, K.: Optimizing costly functions with simple constraints: A limited-memory projected quasi-newton algorithm. In: AI & Statistics (2009)

[23] Silva, R., Scheine, R., Glymour, C., Spirtes, P.: Learning the structure of linear latent variable models. The Journal of Machine Learning Research **7**, 191–246 (2006)

[24] Stegle, O., Lippert, C., Mooij, J., Lawrence, N., Borgwardt, K.: Efficient inference in matrix-variate gaussian models with iid observation noise. Advances in Neural Information Processing Systems (2011)

[25] Teh, Y., Seeger, M., Jordan, M.: Semiparametric latent factor models. In: AI & Statistics (2005)

[26] Tibshirani, R.: Regression shrinkage and selection via the lasso. Journal of the Royal Statistical Society. Series B (Methodological) (1996)

[27] Wainwright, M., Simoncelli, E.: Scale mixtures of gaussians and the statistics of natural images. Advances in neural information processing systems (2000)

[28] Yuan, M., Lin, Y.: Model selection and estimation in the gaussian graphical model. Biometrika (2007)

